# Image Denoising and Inpainting with Deep Neural Networks

**Junyuan Xie, Linli Xu, Enhong Chen**[1]
School of Computer Science and Technology
University of Science and Technology of China
eric.jy.xie@gmail.com, linlixu@ustc.edu.cn, cheneh@ustc.edu.cn

## Abstract

We present a novel approach to low-level vision problems that combines sparse coding and deep networks pre-trained with denoising auto-encoder (DA). We propose an alternative training scheme that successfully adapts DA, originally designed for unsupervised feature learning, to the tasks of image denoising and blind inpainting. Our method's performance in the image denoising task is comparable to that of KSVD which is a widely used sparse coding technique. More importantly, in blind image inpainting task, the proposed method provides solutions to some complex problems that have not been tackled before. Specifically, we can automatically remove complex patterns like superimposed text from an image, rather than simple patterns like pixels missing at random. Moreover, the proposed method does not need the information regarding the region that requires inpainting to be given a priori. Experimental results demonstrate the effectiveness of the proposed method in the tasks of image denoising and blind inpainting. We also show that our new training scheme for DA is more effective and can improve the performance of unsupervised feature learning.

## 1   Introduction

Observed image signals are often corrupted by acquisition channel or artificial editing. The goal of image restoration techniques is to restore the original image from a noisy observation of it. Image denoising and inpainting are common image restoration problems that are both useful by themselves and important preprocessing steps of many other applications. Image denoising problems arise when an image is corrupted by additive white Gaussian noise which is common result of many acquisition channels, whereas image inpainting problems occur when some pixel values are missing or when we want to remove more sophisticated patterns, like superimposed text or other objects, from the image. This paper focuses on image denoising and blind inpainting.

Various methods have been proposed for image denoising. One approach is to transfer image signals to an alternative domain where they can be more easily separated from the noise [1, 2, 3]. For example, Bayes Least Squares with a Gaussian Scale-Mixture (BLS-GSM), which was proposed by Portilla et al, is based on the transformation to wavelet domain [2].

Another approach is to capture image statistics directly in the image domain. Following this strategy, A family of models exploiting the (linear) sparse coding technique have drawn increasing attention recently [4, 5, 6, 7, 8, 9]. Sparse coding methods reconstruct images from a sparse linear combination of an over-complete dictionary. In recent research, the dictionary is learned from data instead of hand crafted as before. This learning step improves the performance of sparse coding significantly. One example of these methods is the KSVD sparse coding algorithm proposed in [6].

Image inpainting methods can be divided into two categories: non-blind inpainting and blind inpainting. In non-blind inpainting, the regions that need to be filled in are provided to the algorithm a priori, whereas in blind inpainting, no information about the locations of the corrupted pixels is given and the algorithm must automatically identify the pixels that require inpainting. The state-of-the-art non-blind inpainting algorithms can perform very well on removing text, doodle, or even very large objects [10, 11, 12]. Some image denoising methods, after modification, can also be applied to non-blind image inpainting with state-of-the-art results [7]. Blind inpainting, however, is a much harder problem. To the best of our knowledge, existing algorithms can only address i.i.d. or simply structured impulse noise [13, 14, 15].

Although sparse coding models perform well in practice, they share a shallow linear structure. Recent research suggests, however, that non-linear, deep models can achieve superior performance in various real world problems. One typical category of deep models are multi-layer neural networks. In [16], Jain et al. proposed to denoise images with convolutional neural networks. In this paper, we propose to combine the advantageous "sparse" and "deep" principles of sparse coding and deep networks to solve the image denoising and blind inpainting problems. The sparse variants of deep neural network are expected to perform especially well in vision problems because they have a similar structure to human visual cortex [17].

Deep neural networks with many hidden layers were generally considered hard to train before a new training scheme was proposed which is to adopt greedy layer-wise pre-training to give better initialization of network parameters before traditional back-propagation training [18, 19]. There exist several methods for pre-training, including Restricted Boltzmann Machine (RBM) and Denoising Auto-encoder (DA) [20, 21].

We employ DA to perform pre-training in our method because it naturally lends itself to denoising and inpainting tasks. DA is a two-layer neural network that tries to reconstruct the original input from a noisy version of it. The structure of a DA is shown in Fig.1a. A series of DAs can be stacked to form a deep network called Stacked Denoising Auto-encoders (SDA) by using the hidden layer activation of the previous layer as input of the next layer.

SDA is widely used for unsupervised pre-training and feature learning [21]. In these settings, only the clean data is provided while the noisy version of it is generated during training by adding random Gaussian or Salt-and-Pepper noise to the clean data. After training of one layer, only the clean data is passed on to the network to produce the clean training data for the next layer while the noisy data is discarded. The noisy training data for the next layer is similarly constructed by randomly corrupting the generated clean training data.

For the image denoising and inpainting tasks, however, the choices of clean and noisy input are natural: they are set to be the desired image after denoising or inpainting and the observed noisy image respectively. Therefore, we propose a new training scheme that trains the DA to reconstruct the clean image from the corresponding noisy observation. After training of the first layer, the hidden layer activations of both the noisy input and the clean input are calculated to serve as the training data of the second layer. Our experiments on the image denoising and inpainting tasks demonstrate that SDA is able to learn features that adapt to specific noises from white Gaussian noise to superimposed text.

Inspired by SDA's ability to learn noise specific features in denoising tasks, we argue that in unsupervised feature learning problems the type of noise used can also affect the performance. Specifically, instead of corrupting the input with arbitrarily chosen noise, more sophisticated corruption process that agrees to the true noise distribution in the data can improve the quality of the learned features. For example, when learning audio features, the variations of noise on different frequencies are usually different and sometimes correlated. Hence instead of corrupting the training data with simple i.i.d. Gaussian noise, Gaussian noise with more realistic parameters that are either estimated from data or suggested by theory should be a better choice.

## 2   Model Description

In this section, we first introduce the problem formulation and some basic notations. Then we briefly give preliminaries about Denoising Auto-encoder (DA), which is a fundamental building block of our proposed method.

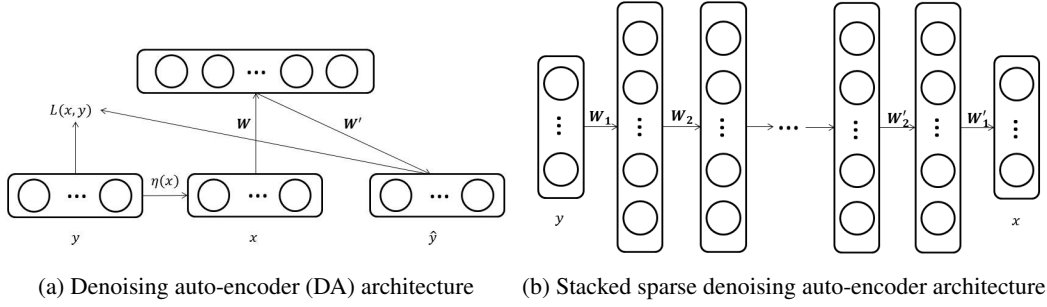

(a) Denoising auto-encoder (DA) architecture      (b) Stacked sparse denoising auto-encoder architecture

Figure 1: Model architectures.

## 2.1 Problem Formulation

Assuming $\mathbf{x}$ is the observed noisy image and $\mathbf{y}$ is the original noise free image, we can formulate the image corruption process as:

$$\mathbf{x} = \eta(\mathbf{y}). \tag{1}$$

where $\eta : \mathbb{R}^n \to \mathbb{R}^n$ is an arbitrary stochastic corrupting process that corrupts the input. Then, the denoising task's learning objective becomes:

$$f = \underset{f}{\arg\min}\, \mathbf{E_y} \| f(\mathbf{x}) - \mathbf{y} \|_2^2 \tag{2}$$

From this formulation, we can see that the task here is to find a function $f$ that best approximates $\eta^{-1}$. We can now treat the image denoising and inpainting problems in a unified framework by choosing appropriate $\eta$ in different situations.

## 2.2 Denoising Auto-encoder

Let $\mathbf{y_i}$ be the original data for $i = 1, 2, ..., N$ and $\mathbf{x_i}$ be the corrupted version of corresponding $\mathbf{y_i}$. DA is defined as shown in Fig.1a:

$$\mathbf{h(x_i)} \;=\; \sigma(\mathbf{Wx_i} + \mathbf{b}) \tag{3}$$
$$\mathbf{\hat{y}(x_i)} \;=\; \sigma(\mathbf{W'h(x_i)} + \mathbf{b'}) \tag{4}$$

where $\sigma(x) = (1 + \exp(-x))^{-1}$ is the sigmoid activation function which is applied element-wise to vectors, $\mathbf{h_i}$ is the hidden layer activation, $\mathbf{\hat{y}(x_i)}$ is an approximation of $\mathbf{y_i}$ and $\mathbf{\Theta} = \{\mathbf{W}, \mathbf{b}, \mathbf{W'}, \mathbf{b'}\}$ represents the weights and biases. DA can be trained with various optimization methods to minimize the reconstruction loss:

$$\theta = \underset{\theta}{\arg\min} \sum_{i=1}^{N} \| \mathbf{y_i} - \mathbf{\hat{y}(x_i)} \|. \tag{5}$$

After finish training a DA, we can move on to training the next layer by using the hidden layer activation of the first layer as the input of the next layer. This is called Stacked denoising auto-encoder (SDA) [21].

## 2.3 Stacked Sparse Denoising Auto-encoders

In this section, we will describe the structure and optimization objective of the proposed model Stacked Sparse Denoising Auto-encoders (SSDA). Due to the fact that directly processing the entire image is intractable, we instead draw overlapping patches from the image as our data objects. In the training phase, the model is supplied with both the corrupted noisy image patches $\mathbf{x_i}$, for $i = 1, 2, ..., N$, and the original patches $\mathbf{y_i}$. After training, SSDA will be able to reconstruct the corresponding clean image given any noisy observation.

To combine the virtues of sparse coding and neural networks and avoid over-fitting, we train a DA to minimize the reconstruction loss regularized by a sparsity-inducing term:

$$L_1(\mathbf{X}, \mathbf{Y}; \theta) = \frac{1}{N} \sum_{i=1}^{N} \frac{1}{2} \| \mathbf{y_i} - \mathbf{\hat{y}(x_i)} \|_2^2 + \beta\, \mathrm{KL}(\hat{\rho} \| \rho) + \frac{\lambda}{2} (\| \mathbf{W} \|_F^2 + \| \mathbf{W'} \|_F^2) \tag{6}$$

| Method | Standard deviation $\sigma$ | | |
|---|---|---|---|
| | 25/PSNR=20.17 | 50/PSNR=14.16 | 100/PSNR=8.13 |
| SSDA | $30.52 \pm 1.02$ | $27.37 \pm 1.10$ | $24.18 \pm 1.39$ |
| BLS-GSM | $30.49 \pm 1.17$ | $27.28 \pm 1.44$ | $24.37 \pm 1.36$ |
| KSVD | $30.96 \pm 0.77$ | $27.34 \pm 1.11$ | $23.50 \pm 1.15$ |

Table 1: Comparison of the denoising performance. Performance is measured by Peak Signal to Noise Ratio (PSNR). Results are averaged over testing set.

where

$$\mathrm{KL}(\hat{\rho}\|\rho) = \sum_{j=1}^{|\hat{\rho}|} \rho \log \frac{\rho}{\hat{\rho}_j} + (1-\rho) \log \frac{(1-\rho)}{1-\hat{\rho}_j}, \qquad \hat{\rho} = \frac{1}{N} \sum_{i}^{N} \mathbf{h}(\mathbf{x_i}).$$

and $\mathbf{h}(\cdot)$ and $\hat{\mathbf{y}}(\cdot)$ are defined in (3), (4) respectively. Here $\hat{\rho}$ is the average activation of the hidden layer. We regularize the hidden layer representation to be sparse by choosing small $\rho$ so that the KL-divergence term will encourage the mean activation of hidden units to be small. Hence the hidden units will be zero most of the time and achieve sparsity.

After training of the first DA, we use $\mathbf{h}(\mathbf{y_i})$ and $\mathbf{h}(\mathbf{x_i})$ as the clean and noisy input respectively for the second DA. This is different from the approach described in [21], where $\mathbf{x_i}$ is discarded and $\eta(\mathbf{h}(\mathbf{y_i}))$ is used as the noisy input. We point out that our method is more natural in that, since $\mathbf{h}(\mathbf{y_i})$ lies in a different space from $\mathbf{y_i}$, the meaning of applying $\eta(\cdot)$ to $\mathbf{h}(\mathbf{y_i})$ is not clear.

We then initialize a deep network with the weights obtained from $K$ stacked DAs. The network has one input layer, one output and $2K - 1$ hidden layers as shown in Fig.1b. The entire network is then trained using the standard back-propagation algorithm to minimize the following objective:

$$L_2(\mathbf{X}, \mathbf{Y}; \theta) = \frac{1}{N} \sum_{i=1}^{N} \frac{1}{2} \|\mathbf{y_i} - \mathbf{y}(\mathbf{x_i})\|_2^2 + \frac{\lambda}{2} \sum_{j=1}^{2K} (\|\mathbf{W_j}\|_F^2). \tag{7}$$

Here we removed the sparsity regularization because the pre-trained weights will serve as regularization to the network [18].

In both of the pre-training and fine-tuning stages, the loss functions are optimized with L-BFGS algorithm (a Quasi-Newton method) which, according to [22], can achieve fastest convergence in our settings.

## 3 Experiments

We narrow our focus down to denoising and inpainting of grey-scale images, but there is no difficulty in generalizing to colored images. We use a set of natural images collected from the web[1] as our training set and standard testing images[2] as the testing set. We create noisy images from clean training and testing images by applying the function (1) to them. Image patches are then extracted from both clean and noisy images to train SSDAs. We employ Peak Signal to Noise Ratio (PSNR) to quantify denoising results: $10 \log_{10}(255^2/\sigma_e^2)$, where $\sigma_e^2$ is the mean squared error. PSNR is one of the standard indicators used for evaluating image denoising results.

### 3.1 Denoising White Gaussian Noise

We first corrupt images with additive white Gaussian noise of various standard deviations. For the proposed method, one SSDA model is trained for each noise level. We evaluate different hyper-parameter combinations and report the best result. We set $K$ to 2 for all cases because adding more layers may slightly improve the performance but require much more training time. In the meantime, we try different patch sizes and find that higher noise level generally requires larger patch size. The

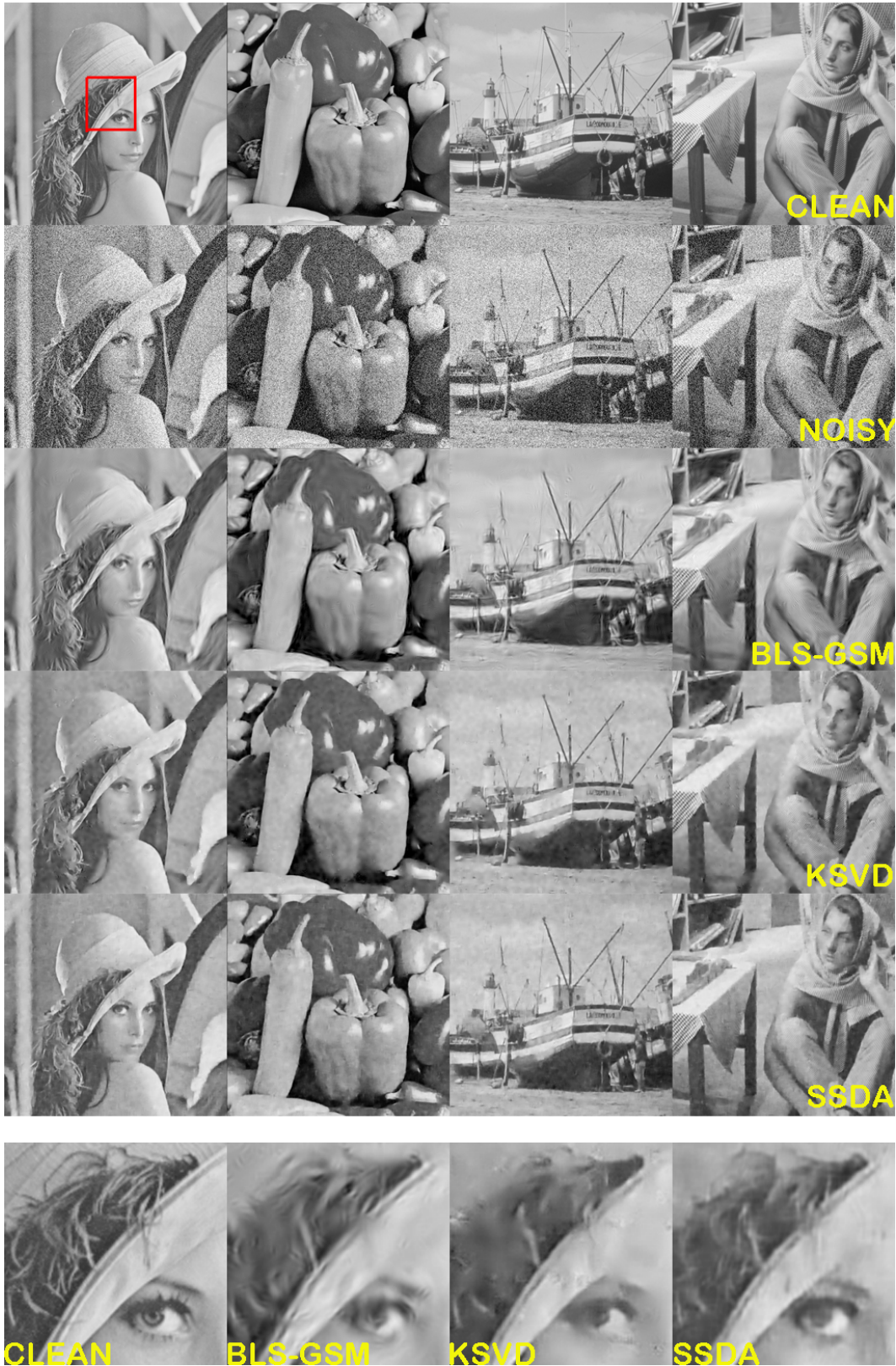

Figure 2: Visual comparison of denoising results. Results of images corrupted by white Gaussian noise with standard deviation $\sigma = 50$ are shown. The last row zooms in on the outlined region of the original image.

dimension of hidden layers is generally set to be a constant factor times the dimension of the input[3]. SSDA is not very sensitive to the weights of the regularization terms. For Bayes Least Squares-Gaussian Scale Mixture (BLS-GSM) and KSVD method, we use the fully trained and optimized toolbox obtained from the corresponding authors [2, 7]. All three models are tuned to specific noise level of each input. The comparison of quantitative results are shown in Tab.1. Numerical results showed that differences between the three algorithms are statistical insignificant. A visual comparison is shown in Fig.2. We find that SSDA gives clearer boundary and restores more texture details than KSVD and BLS-GSM although the PSNR scores are close. This indicates that although the reconstruction errors averaged over all pixels are the same, SSDA is better at denoising complex regions.

## 3.2    Image Inpainting

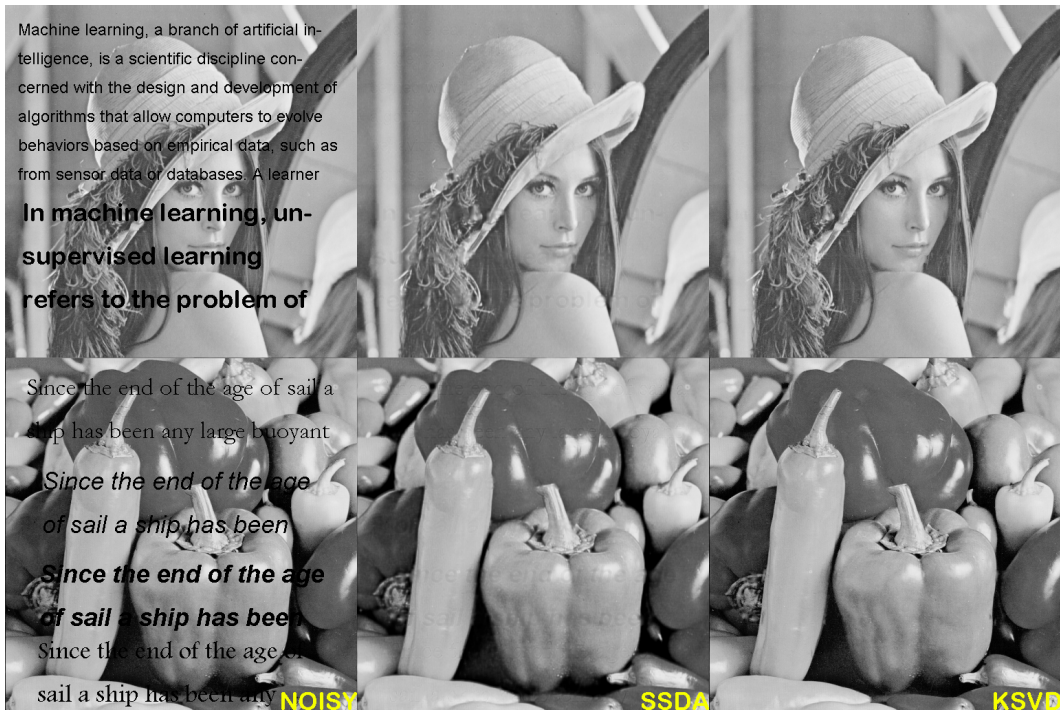

Figure 3: Visual comparison of inpainting results.

For the image inpainting task, we test our model on the text removal problem. Both the training and testing set compose of images with super-imposed text of various fonts and sizes from 18-pix to 36-pix. Due to the lack of comparable blind inpainting algorithms, We compare our method to the non-blind KSVD inpainting algorithm [7], which significantly simplifies the problem by requiring the knowledge of which pixels are corrupted and require inpainting. A visual comparison is shown in Fig.3. We find that SSDA is able to eliminate text of small fonts completely while text of larger fonts is dimmed. The proposed method, being blind, generates results comparable to KSVD's even though KSVD is a non-blind algorithm. Non-blind inpainting is a well developed technology that works decently on the removal of small objects. Blind inpainting, however, is much harder since it demands automatic identification of the patterns that requires inpainting, which, by itself is a very challenging problem. To the best of our knowledge, former methods are only capable of removing i.i.d. or simply structured impulse noise [13, 14, 15]. SSDA's capability of blind inpainting of complex patterns is one of this paper's major contributions.

| Training noise | Testing noise | | |
|---|---|---|---|
| | Gaussian | Salt-and-Pepper | Image background |
| Gaussian | **91.42%** | 82.95% | 86.45% |
| Salt-and-Pepper | 90.05% | **90.14%** | 81.77% |
| Image background | 84.88% | 74.47% | **86.87%** |

Table 2: Comparison of classification results. Highest accuracy in each column is shown in bold font.

### 3.3 Hidden Layer Feature Analysis

Traditionally when training denoising auto-encoders, the noisy training data is usually generated with arbitrarily selected simple noise distribution regardless of the characteristics of the specific training data [21]. However, we propose that this process deserves more attention. In real world problems, the clean training data is in fact usually subject to noise. Hence, if we estimate the distribution of noise and exaggerate it to generate noisy training data, the resulting DA will learn to be more robust to noise in the input data and produce better features.

Inspired by SSDA's ability to learn different features when trained on denoising different noise patterns, we argue that training denoising auto-encoders with noise patterns that fit to specific situations can also improve the performance of unsupervised feature learning. We demonstrate this by a comparison of classification performance with different sets of features learned on the MNIST dataset. We train DAs with different types of noise and then apply them to handwritten digits corrupted by the type of noise they are trained on as well as other types of noise. We compare the quality of the learned features by feeding them to SVMs and comparing the corresponding classification accuracy. The results are shown in Tab.2. We find that the highest classification accuracy on each type of noise is achieved by the DA trained to remove that type of noise. This is not surprising since more information is utilized, however it indicates that instead of arbitrarily corrupting input with noise that follows simple distribution and feeding it to DA, more sophisticated methods that corrupt input in more realistic ways can achieve better performance.

## 4 Discussion

### 4.1 Prior vs. Learned Structure

Unlike models relying on structural priors, our method's denoising ability comes from learning. Some models, for example BLS-GSM, have carefully designed structures that can give surprisingly good results with random parameter settings [23]. However, randomly initialized SSDA obviously can not produce any meaningful results. Therefore SSDA's ability to denoise and inpaint images is mostly the result of training. Whereas models that rely on structural priors usually have very limited scope of applications, our model can be adapted to other tasks more conveniently. With some modifications, it is possible to denoise audio signals or complete missing data (as a data preprocessing step) with SSDA.

### 4.2 Advantages and Limitations

Traditionally, for complicated inpainting tasks, an inpainting mask that tells the algorithm which pixels correspond to noise and require inpainting is supplied a priori. However, in various situations this is time consuming or sometimes even impossible. Our approach, being blind, has significant advantages in such circumstances. This makes our method a suitable choice for fully automatic and noise pattern specific image processing.

The limitation of our method is also obvious: SSDA strongly relies on supervised training. In our experiment, we find that SSDA can generalize to unseen, but similar noise patterns. Generally speaking, however, SSDA can remove only the noise patterns it has seen in the training data. Therefore,

SSDA would only be suitable in circumstances where the scope of denoising tasks is narrow, such as reconstructing images corrupted by a certain procedure.

## 5    Conclusion

In this paper, we present a novel approach to image denoising and blind inpainting that combines sparse coding and deep neural networks pre-trained with denoising auto-encoders. We propose a new training scheme for DA that makes it possible to denoise and inpaint images within a unified framework. In the experiments, our method achieves performance comparable to traditional linear sparse coding algorithm on the simple task of denoising additive white Gaussian noise. Moreover, our non-linear approach successfully tackles the much harder problem of blind inpainting of complex patterns which, to the best of our knowledge, has not been addressed before. We also show that the proposed training scheme is able to improve DA's performance in the tasks of unsupervised feature learning.

In our future work, we would like to explore the possibility of adapting the proposed approach to various other applications such as denoising and inpainting of audio and video, image super-resolution and missing data completion. It is also meaningful to investigate into the effects of different hyper-parameter settings on the learned features.

## 6    Acknowledgement

Research supported by grants from the National Natural Science Foundation of China (No. 61003135 & No. 61073110), NSFC Major Program (No. 71090401/71090400), the Fundamental Research Funds for the Central Universities (WK0110000022), the National Major Special Science & Technology Projects (No. 2011ZX04016-071), and Research Fund for the Doctoral Program of Higher Education of China (20093402110017, 20113402110024).

## Footnotes

[1]Corresponding author.

[1]http://decsai.ugr.es/cvg/dbimagenes/

[2]Widely used images commonly referred to as Lena, Barbara, Boat, Pepper, etc. in the image processing community.

[3]We set this factor to 5. The other hyper-parameters are: $\lambda = 10^{-4}, \beta = 10^{-2}, \rho = 0.05$.

## References

[1] J. Xu, K. Zhang, M. Xu, and Z. Zhou. An adaptive threshold method for image denoising based on wavelet domain. *Proceedings of SPIE, the International Society for Optical Engineering*, 7495:165, 2009.

[2] J. Portilla, V. Strela, M.J. Wainwright, and E.P. Simoncelli. Image denoising using scale mixtures of Gaussians in the wavelet domain. *Image Processing, IEEE Transactions on*, 12(11):1338–1351, 2003.

[3] F. Luisier, T. Blu, and M. Unser. A new SURE approach to image denoising: Interscale orthonormal wavelet thresholding. *IEEE Transactions on Image Processing*, 16(3):593–606, 2007.

[4] B.A. Olshausen and D.J. Field. Sparse coding with an overcomplete basis set: A strategy employed by V1? *Vision research*, 37(23):3311–3325, 1997.

[5] K. Kreutz-Delgado, J.F. Murray, B.D. Rao, K. Engan, T.W. Lee, and T.J. Sejnowski. Dictionary learning algorithms for sparse representation. *Neural computation*, 15(2):349–396, 2003.

[6] M. Elad and M. Aharon. Image denoising via sparse and redundant representations over learned dictionaries. *IEEE Transactions on Image Processing*, 15(12):3736–3745, 2006.

[7] J. Mairal, M. Elad, and G. Sapiro. Sparse representation for color image restoration. *IEEE Transactions on Image Processing*, 17(1):53–69, 2008.

[8] X. Lu, H. Yuan, P. Yan, Y. Yuan, L. Li, and X. Li. Image denoising via improved sparse coding. *Proceedings of the British Machine Vision Conference*, pages 74–1, 2011.

[9] J. Mairal, F. Bach, J. Ponce, and G. Sapiro. Online dictionary learning for sparse coding. *Proceedings of the 26th Annual International Conference on Machine Learning*, pages 689–696, 2009.

[10] A. Criminisi, P. Pérez, and K. Toyama. Region filling and object removal by exemplar-based image inpainting. *IEEE Transactions on Image Processing*, 13(9):1200–1212, 2004.

[11] M. Bertalmio, G. Sapiro, V. Caselles, and C. Ballester. Image inpainting. *Proceedings of the 27th annual conference on Computer graphics and interactive techniques*, pages 417–424, 2000.

[12] A. Telea. An image inpainting technique based on the fast marching method. *Journal of graphics tools.*, 9(1):23–34, 2004.

[13] B. Dong, H. Ji, J. Li, Z. Shen, and Y. Xu. Wavelet frame based blind image inpainting. *Applied and Computational Harmonic Analysis*, 2011.

[14] Y. Wang, A. Szlam, and G. Lerman. Robust locally linear analysis with applications to image denoising and blind inpainting. *preprint*, 2011.

[15] M. Yan. Restoration of images corrupted by impulse noise using blind inpainting and l0 norm. *preprint*, 2011.

[16] V. Jain and H.S. Seung. Natural image denoising with convolutional networks. *Advances in Neural Information Processing Systems*, 21:769–776, 2008.

[17] H. Lee, C. Ekanadham, and A. Ng. Sparse deep belief net model for visual area V2. *Advances in Neural Information Processing Systems 20*, pages 873–880, 2008.

[18] D. Erhan, Y. Bengio, A. Courville, P.A. Manzagol, P. Vincent, and S. Bengio. Why does unsupervised pre-training help deep learning? *The Journal of Machine Learning Research*, 11:625–660, 2010.

[19] Y. Bengio. Learning deep architectures for AI. *Foundations and Trends® in Machine Learning*, 2(1):1–127, 2009.

[20] R. Salakhutdinov and G.E. Hinton. Deep boltzmann machines. *Proceedings of the international conference on artificial intelligence and statistics*, 5(2):448–455, 2009.

[21] P. Vincent, H. Larochelle, I. Lajoie, Y. Bengio, and P.A. Manzagol. Stacked denoising autoencoders: Learning useful representations in a deep network with a local denoising criterion. *The Journal of Machine Learning Research*, 11:3371–3408, 2010.

[22] Q.V. Le, A. Coates, B. Prochnow, and A.Y. Ng. On optimization methods for deep learning. *Learning*, pages 265–272, 2011.

[23] S. Roth and M.J. Adviser-Black. High-order markov random fields for low-level vision. *Brown University Press*, 2007.

